# Viewing Classifier Systems
# as Model Free Learning in POMDPs

**Akira Hayashi and Nobuo Suematsu**
Faculty of Information Sciences
Hiroshima City University
3-4-1 Ozuka-higashi, Asaminami-ku, Hiroshima, 731-3194 Japan
{akira,suematsu}@im.hiroshima-cu.ac.jp

## Abstract

Classifier systems are now viewed disappointing because of their problems such as the rule strength vs rule set performance problem and the credit assignment problem. In order to solve the problems, we have developed a hybrid classifier system: GLS (Generalization Learning System). In designing GLS, we view CSs as model free learning in POMDPs and take a hybrid approach to finding the best generalization, given the total number of rules. GLS uses the policy improvement procedure by Jaakkola et al. for an locally optimal stochastic policy when a set of rule conditions is given. GLS uses GA to search for the best set of rule conditions.

## 1 INTRODUCTION

Classifier systems (CSs) (Holland 1986) have been among the most used in reinforcement learning. Some of the advantages of CSs are (1) they have a built-in feature (the use of don't care symbols "#") for input generalization, and (2) the complexity of policies can be controlled by restricting the number of rules. In spite of these attractive features, CSs are now viewed somewhat disappointing because of their problems (Wilson and Goldberg 1989; Westerdale 1997). Among them are the rule strength vs rule set performance problem, the definition of the rule strength parameter, and the credit assignment (BBA vs PSP) problem.

In order to solve the problems, we have developed a hybrid classifier system: GLS (Generalization Learning System). GLS is based on the recent progress of RL research in partially observable Markov decision processes (POMDPs). In POMDPs, the environments are really Markovian, but the agent cannot identify the state from the current observation. It may be due to noisy sensing or *perceptual aliasing*. Perceptual aliasing occurs when the sensor returns the same observation in multiple states. Note that even for a completely observable

MDP, the use of don't care symbols for input generalization will make the process as if it were partially observable.

In designing GLS, we view CSs as RL in POMDPs and take a hybrid approach to finding the best generalization, given the total number of rules. GLS uses the policy improvement procedure in Jaakkola et al. (1994) for an locally optimal stochastic policy when a set of rule conditions is given. GLS uses GA to search for the best set of rule conditions.

The paper is organized as follows. Since CS problems are easier to understand from GLS perspective, we introduce Jaakkola et al. (1994), propose GLS, and then discuss CS problems.

## 2  LEARNING IN POMDPS

Jaakkola et al. (1994) consider POMDPs with perceptual aliasing and memoryless stochastic policies. Following the authors, let us call the observations *messages*. Therefore, a policy is a mapping from messages to probability distributions (PDs) over the actions.

Given a policy $\pi$, the value of a state $s$, $V^\pi(s)$, is defined for POMDPs just as for MDPs. Then, the value of a message $m$ under policy $\pi$, $V^\pi(m)$, can be defined as follows:

$$V^\pi(m) = \sum_{s \in S} P^\pi(s|m) V^\pi(s) \tag{1}$$

where $P^\pi(s|m)$ is the probability that the state is $s$ when the message is $m$ under the policy $\pi$.

Then, the following holds.

$$V^\pi(s) = \lim_{N \to \infty} \sum_{t=1}^{N} E\{R(s_t, a_t) - \overline{R} \mid s_1 = s\} \tag{2}$$

$$V^\pi(m) = E\{V(s) \mid s \to m\} \tag{3}$$

where $s_t$ and $a_t$ refer to the state and the action taken at the $t^{th}$ step respectively, $R(s_t, a_t)$ is the immediate reward at the $t^{th}$ step, $\overline{R}$ is the (unknown) gain (i.e. the average reward per step). $s \to m$ refers to all the instances where $m$ is observed in $s$ and $E\{\cdot \mid s \to m\}$ is a Monte-Carlo expectation.

In order to compute $E\{V(s) \mid s \to m\}$, Jaakkola et al. showed a Monte-Carlo procedure:

$$
\begin{aligned}
V_t^\pi(m) = \frac{1}{k}\{ \quad & R_{t_1} \quad + \Gamma_{1,1} R_{t_1+1} + \Gamma_{1,2} R_{t_1+2} + \cdots + \Gamma_{1,t-t_1} R_t \\
+ \quad & R_{t_2} \quad + \Gamma_{2,1} R_{t_2+1} + \Gamma_{2,2} R_{t_2+2} + \cdots + \Gamma_{2,t-t_2} R_t \\
& \vdots \\
+ \quad & R_{t_k} \quad + \Gamma_{k,1} R_{t_k+1} + \cdots + \Gamma_{k,t-t_k} R_t \}
\end{aligned}
\tag{4}
$$

where $t_k$ denotes the time step corresponding to the $k^{th}$ occurrence of the message $m$, $R_t = R(s_t, a_t) - \overline{R}$ for every $t$, $\Gamma_{k,T}$ indicates the discounting at the $T^{th}$ step in the $k^{th}$ sequence. By estimating $\overline{R}$ and by suitably setting $\Gamma_{k,T}$, $V_t^\pi(m)$ converges to $V^\pi(m)$. $Q^\pi(m, a)$, Q-value of the message $m$ for the action $a$ under the policy $\pi$, is also defined and computed in the same way.

Jaakkola et al. have developed a policy improvement method:

**Step 1** Evaluate the current policy $\pi$ by computing $V^\pi(m)$ and $Q^\pi(m, a)$ for each $m$ and $a$.

**Step 2** Test for any $m$ whether $\max_a Q^\pi(m,a) > V^\pi(m)$ holds. If *not*, then return $\pi$.

**Step 3** For each $m$ and $a$, define $\pi^1(a|m)$ as follows:
$\pi^1(a|m) = 1.0$ when $a = argmax_a Q^\pi(m,a)$,   $\pi^1(a|m) = 0.0$ otherwise.
Then, define $\pi^\epsilon$ as $\pi^\epsilon(a|m) = (1-\epsilon)\pi(a|m) + \epsilon\pi^1(a|m)$

**Step 4** Set the new policy as $\pi = \pi^\epsilon$, and goto Step1.

## 3   GLS

Each rule in GLS consists of a condition part, an action part, and an evaluation part: *Rule* = (*Condition*, *Action*, *Evaluation*). The condition part is a string $c$ over the alphabet $\{0,1,\#\}$, and is compared with a binary sensor message. $\#$ is a don't care symbol, and matches 0 and 1. When the condition $c$ matches the message, the action is randomly selected using the PD in the action part: *Action* = $(p(a_1|c), p(a_2|c), \ldots, p(a_{|A|}|c)), \sum_{j=1}^{|A|} p(a_j|c) = 1.0$ where $|A|$ is the total number of actions. The evaluation part records the value of the condition $V(c)$ and the Q-values of the condition action pairs $Q(c,a)$: *Evaluation* = $(V(c), Q(c,a_1), Q(c,a_2), \ldots, Q(c,a_{|A|}))$. Each *rule set* consists of $N$ rules, $\{Rule_1, Rule_2, \ldots, Rule_N\}$. $N$, the total number of rules in a rule set, is a design parameter to control the complexity of policies. All the rules except the last one are called *standard* rules. The last rule $Rule_N$ is a special rule which is called the *default* rule. The condition part of the default rule is a string of $\#$'s and matches any message.

Learning in GLS proceeds as follows: (1)Initialization: randomly generate an initial population of $M$ rule sets, (2)Policy Evaluation and Improvement: for each rule set, repeat a policy evaluation and improvement cycle for a suboptimal policy, then, record the gain of the policy for each rule set, (3)Genetic Algorithm: use the gain of each rule set as its fitness measure and produce a new generation of rule sets, (4) Repeat: repeat from the policy evaluation and improvement step with the new generation of rule sets.

In (2)Policy Evaluation and Improvement, GLS repeats the following cycle for each rule set.

**Step 1** Set $\epsilon$ sufficiently small. Set $t^{max}$ sufficiently large.

**Step 2** Repeat for $1 \le t \le t^{max}$.

1. Make an observation of the environment and receive a message $m_t$ from the sensor.

2. From all the rules whose condition matches the message $m_t$, find the rule whose condition is the most *specific*[1]. Let us call the rule the *active* rule.

3. Select the next action $a_t$ randomly according to the PD in the action part of the active rule, execute the action, and receive the reward $R(s_t, a_t)$ from the environment. (The state $s_t$ is not observable.)

4. Update the current estimate of the gain $\overline{R}$ from its previous estimate and $R(s_t, a_t)$. Let $R_t = R(s_t, a_t) - \overline{R}$. For each rule, consider its condition $c_i$ as (a generalization of) a message, and update its evaluation part $V(c_i)$ and $Q(c_i, a)(a \in A)$ using Eq.(4).

**Step 3** Check whether the following holds. If *not*, exit.
$\exists i(1 \le i \le N),$   $\max_a Q(c_i, a) > V(c_i)$

**Step 4** Improve the current policy according to the method in the previous section, and update the action part of the corresponding rules and goto Step 2.

GLS extracts the condition parts of all the rules in a rule set and concatenates them to form a string. The string will be an individual to be manipulated by the genetic algorithm (GA). The genetic algorithm used in GLS is a fairly standard one. GLS combines the SGA (the simple genetic algorithm) (Goldberg 1989) with the elitist keeping strategy. The SGA is composed of three genetic operators: selection, crossover, and mutation. The fitness proportional selection and the single-point crossover are used. The three operators are applied to an entire population at each generation. Since the original SGA does not consider #'s in the rule conditions, we modified SGA as follows. When GLS randomly generates an initial population of rule sets, it generates # at each allele position in rule conditions according to the probability $P_\#$.

## 4   CS PROBLEMS AND GLS

In the history of classifier systems, there were two quite different approaches: the Michigan approach (Holland and Reitman 1978), and the Pittsburgh (Pitt) approach (DeJong 1988). In the Michigan approach, each rule is considered as an individual and the rule set as the population in GA. Each rule has its strength parameter, which is based on its future payoff and is used as the fitness measure in GA. These aspects of the approach cause many problems. One is the rule strength vs rule set performance problem. Can we collect only strong rules and get the best rule set performance? Not necessarily. A strong rule may cooperate with weak rules to increase its payoff. Then, how can we define and compute the strength parameter for the best rule set performance? In spite of its problems, this approach is now so much more popular than the other, that when people simply say classifier systems, they refer to Michigan type classifier systems. In the Pitt approach, the problems of the Michigan approach are avoided by requiring GA to evaluate a whole rule set. In the approach, a rule set is considered as an individual and multiple rule sets are kept as the population. The problem of the Pitt approach is its computational difficulties.

GLS can be considered as a combination of the Michigan and Pitt approaches. GA in GLS works as that in the Pitt approach. It evaluates a total rule set, and completely avoids the rule strength vs rule set performance problem in the Michigan approach. As the Michigan type CSs, GLS evaluates each rule to improve the policy. This alleviates the computational burden in the Pitt approach. Moreover, GLS evaluates each rule in a more formal and sound way than the Michigan approach. The values, $V(c)$, and $Q(c, a)$, are defined on the basis of POMDPs, and the policy improvement procedure using the values is guaranteed to find a local maximum.

Westerdale (1997) has recently made an excellent analysis of problematic behaviors of Michigan type CSs. Two popular methods for credit assignment in CSs are the bucket brigade algorithm (BBA) (Holland 1986) and the profit sharing plan (PSP) (Grefenstette 1988). Westerdale shows that BBA does not work in POMDPs. He insists that PSP with infinite time span is necessary for the right credit assignment, although he does not show how to carry out the computation. GLS does not use BBA or PSP. GLS uses the Monte Carlo procedure, Eq.(4), to compute the value of each condition action pair. The series in Eq.(4) is slow to converge. But, this is the cost we have to pay for the right credit assignment in POMDPs. Westerdale points out another CS problem. He claims that a distinction must be made between the *availability* and the payoff of rules. We agree with him. As he says, if the expected payoff of Rule 1 is twice as much as Rule 2, then we want to *always* choose Rule 1. GLS makes the distinction. The probability of a stochastic policy $\pi(a|c)$ in GLS corresponds to the availability, and the value of a condition action pair $Q(c, a)$ corresponds to the payoff.

Samuel System (Grefenstette et al. 1990) can also be considered as a combination of the Michigan and Pitt approaches. Samuel is a highly sophisticated system which has lots of features. We conjecture, however, that Samuel is not free from the CS problems which

Westerdale has analyzed. This is because Samuel uses PSP for credit assignment, and Samuel uses the payoff of each rule for action selection, and does not make a distinction between the availability and the payoff of rules.

XCS (Wilson 1995) seems to be an exceptionally reliable Michigan-type CS. In XCS, each rule's fitness is based *not* on its future payoff *but* on the prediction accuracy of its future payoff (XCS uses BBA for credit assignment). Wilson reports that XCS's population tends to form a complete and accurate mapping from sensor messages and actions to payoff predictions. We conjecture that XCS tries to build the most general Markovian model of the environment. Therefore, it will be difficult to apply XCS when the environment is not Markovian, or when we cannot afford the number of rules enough to build a Markovian model of the environment, even if the environment itself is Markovian. As we will see in the next section, GLS is intended exactly for these situations.

Kaelbling et al. (1996) surveys methods for input generalization when reward is delayed. The methods use a function approximator to represent the value function by mapping a state description to a value. Since they use value iteration or Q-learning anyway, it is difficult to apply the methods when the generalization violates the Markov assumption and induces a POMDP.

## 5  EXPERIMENTS

We have tested GLS with some of the representative problems in CS literature. Fig. 1 shows Gref1 world (Grefenstette 1987). In Gref1 world, we used GLS to find the smallest rule set which is necessary for the optimal performance. Since this is not a POMDP but an MDP, the optimal policy can easily be learned when we have a corresponding rule for each of the 16 states. However, when the total number of rules is less than that of states, the environment looks like a POMDP to the learning agent, even if the environment itself is an MDP. The graph shows how the gain of the best rule set in the population changes with the generation. We can see from the figure that four rules are enough for the optimal performance. Also note that the saving of the rules is achieved by selecting the most specific matching rule as an active rule. The rule set with this rule selection is called the *default hierarchy* in CS literature.

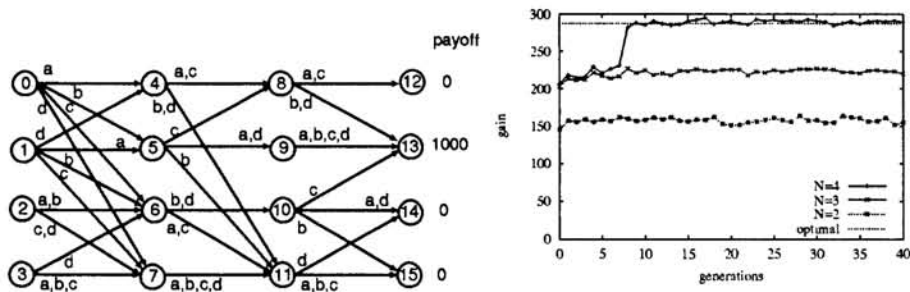

Figure 1: LEFT: GREF1 World. States $\{0, 1, 2, 3\}$ are the start states and states $\{12, 13, 14, 15\}$ are the end states. In each state, the agent gets the state number (4 bits) as a message, and chooses an action a,b,c, or d. When the agent reaches the end states, he receives reward 1000 in state 13, but reward 0 in other states. Then the agent is put in one of the start states with equal probability. We added 10% action errors to make the process ergodic. When an action error occurs, the agent moves to one of the 16 states with equal probability.
RIGHT: Gain of the best rule set. Parameters: $t^{max} = 10000, \epsilon = 0.10, M = 10, N = 2, 3, 4, P_{\#} = 0.33$. For $N = 4$, the best rule set at the $40^{th}$ generation was { if 0101 (State 5) then a 1.0, if 1010 (State 10) then c 1.0, if ##11 (States 3,7,11,15) then d 1.0, if #### (Default Rule) then b 1.0}.

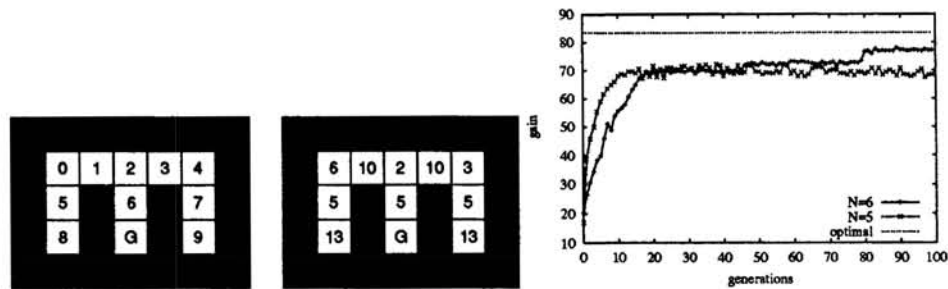

Figure 2: LEFT: McCallum's Maze. We show the state numbers in the left, and the messages in the right. States 8 and 9 are the start states, and state G is the goal state. In each state, the agent receives a sensor message which is 4 bit long, Each bit in the message tells whether a wall exists in each of the four directions. From each state, the agent moves to one of the adjacent states. When the agent reaches the goal state, he receives reward 1000. The agent is then put in one of the start states with equal probability.
RIGHT: Gain of the best rule set. Parameters: $t^{max} = 50000, \epsilon = 0.10, M = 10, N = 5, 6, P_\# = 0.33$.

Fig. 2 is a POMDP known as as McCallum's Maze (McCallum 1993). Thanks to the use of stochastic policies, GLS achieves near optimal gain for memoryless policies. Note that no memoryless deterministic policy can take the agent to the goal for this problem.

We have seen GLS's generalization capability for an MDP in Gref1 World, the advantage of stochastic policies for a POMDP in McCallum's maze. In Woods7 (Wilson 1994), we attempt to test GLS's generalization capability for a POMDP. See Fig. 3. Since each sensor message is 16 bit long, and the conditions of GLS rules can have either 0,1,or # for each of the 16 bits, there are $3^{16}$ possible conditions in total. When we notice that there are only 92 different actual sensor messages in the environment, it seems quite difficult to discover them only by using GA. In fact, when we ran GLS for the first time, the standard rules very rarely matched the messages and the default rule took over most of the time. In order to avoid the no matching rule problem, we made the number of rules in a rule set large ($N = 100$), increased $P_\#$ from 0.33 in the previous problems to 0.70.

The problem was independently attacked by other methods. Wilson applied his ZCS, zeroth level classifier system, to Woods7 (Wilson 1994). The gain was 0.20. ZCS has a special covering procedure to turn around the no matching rule problem. The covering procedure generates a rule which matches a message when none of the current rules matches the message. We expect further improvement on the gain, if we equip GLS with some covering procedure.

## 6  SUMMARY

In order to solve the CS problems such as the rule strength vs rule set performance problem and the credit assignment problem, we have developed a hybrid classifier system: GLS. We notice that generalization often leads to state aliasing. Therefore, in designing GLS, we view CSs as model free learning in POMDPs and take a hybrid approach to finding the best generalization, given the total number of rules. GLS uses the policy improvement procedure by Jaakkola et al. for an locally optimal stochastic policy when a set of rule conditions is given. GLS uses GA to search for the best set of rule conditions.

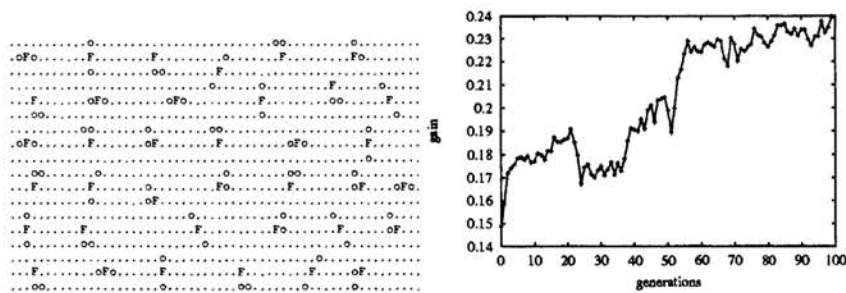

Figure 3: LEFT: Woods7.Each cell is either empty "."., contains a stone "o", or contains food "F". The cells which contain a stone are not passable, and the cells which contain food are goals. In each cell, the agent receives a $2 * 8 = 16$ bit long sensor message, which tells the contents of the eight adjacent cells. From each cell, the agent can move to one of the eight adjacent cells. When the agent reaches a cell which contains food, he receives reward 1. The agent is then put in one of the empty cells with equal probability.
RIGHT:Gain of the best rule set. Parameters: $t^{max} = 10000, \epsilon = 0.10, M = 10, N = 100, P_\# = 0.70$.

## Footnotes

[1]The most specific rule has the least number of $\#$'s. This is intended only for saving the number of rules.

# References

DeJong, K. A. (1988). Learning with genetic algorithms: An overview. *Machine Learning*, 3:121–138.

Goldberg, D. E. (1989). *Genetic Algorithms in Search, Optimization, and Machine Learning*. Addison-Wesley.

Grefenstette, J. J. (1987). Multilevel credit assignment in a genetic learning system. In *Proc. Second Int. Conf. on Genetic Algorithms*, pp. 202–209.

Grefenstette, J. J. (1988). Credit assignment in rule discovery systems based on genetic algorithms. *Machine Learning*, 3:225–245.

Grefenstette, J. J., C. L. Ramsey, and A. C. Schultz (1990). Learning sequential decision rules using simulation and competition. *Machine Learning*, 5:355–381.

Holland, J. H. (1986). Escaping brittleness: the possibilities of general purpose learning algorithms applied to parallel rule-based systems. In *Machine Learning II*, pp. 593–623. Morgan Kaufmann.

Holland, J. H. and J. S. Reitman (1978). Cognitive systems based on adaptive algorithms. In D. A. Waterman and F. Hayes-Roth (Eds.), *Pattern-directed inference systems*. Academic Press.

Jaakkola, T., S. P. Singh, and M. I. Jordan (1994). Reinforcement learning algorithm for partially observable markov decision problems. In *Advances of Neural Information Processing Systems 7*, pp. 345–352.

Kaelbling, L. P., M. L. Littman, and A. W. Moore (1996). Reinforcement learning: A survey. *Journal of Artificial Intelligence Research*, 4:237–285.

McCallum, R. A. (1993). Overcoming incomplete perception with utile distinction memory. In *Proc. the Tenth Int. Conf. on Machine Learning*, pp. 190–196.

Westerdale, T. H. (1997). Classifier systems - no wonder they don't work. In *Proc. Second Annual Genetic Programming Conference*, pp. 529–537.

Wilson, S. W. (1994). Zcs: A zeroth order classifier system. *Evolutionary Computation*, 2(1):1–18.

Wilson, S. W. (1995). Classifier fitness based on accuracy. *Evolutionary Computation*, 3(2):149–175.

Wilson, S. W. and D. E. Goldberg (1989). A critical review of classifier systems. In *Proc. Third Int. Conf. on Genetic Algorithms*, pp. 244–255.
